# Information Bottleneck for Non Co-Occurrence Data

**Yevgeny Seldin**[†]  **Noam Slonim**[*]  **Naftali Tishby**[†‡]

[†]School of Computer Science and Engineering
[‡]Interdisciplinary Center for Neural Computation
The Hebrew University of Jerusalem
[*]The Lewis-Sigler Institute for Integrative Genomics
Princeton University
{seldin,tishby}@cs.huji.ac.il, nslonim@princeton.edu

## Abstract

We present a general model-independent approach to the analysis of data in cases when these data do not appear in the form of co-occurrence of two variables $X, Y$, but rather as a sample of values of an unknown (stochastic) function $Z(X, Y)$. For example, in gene expression data, the expression level $Z$ is a function of gene $X$ and condition $Y$; or in movie ratings data the rating $Z$ is a function of viewer $X$ and movie $Y$. The approach represents a consistent extension of the Information Bottleneck method that has previously relied on the availability of co-occurrence statistics. By altering the relevance variable we eliminate the need in the sample of joint distribution of all input variables. This new formulation also enables simple MDL-like model complexity control and prediction of missing values of $Z$. The approach is analyzed and shown to be on a par with the best known clustering algorithms for a wide range of domains. For the prediction of missing values (collaborative filtering) it improves the currently best known results.

## 1  Introduction

In the situation of information explosion that characterizes todays world, the need for automatic tools for data analysis is more than obvious. Here, we focus on an unsupervised analysis of data that can be organized in matrix form. Clearly, this broad definition covers various types of data. For instance, in text analysis data, the rows of a matrix correspond to words, the columns to different documents, and entries indicate the number of occurrences of a particular word in a specific document. In a matrix of gene expression data, rows correspond to genes, columns to various experimental conditions, and entries indicate expression levels of given genes in given conditions. In movie rating data, rows correspond to viewers, columns to movies, and entries indicate ratings made by the viewers. Finally, for financial data, rows correspond to stocks, columns to different time points, and each entry indicates a price change of a particular stock at a given time point.

While the text analysis case is a classical example of co-occurrence data, the remaining examples are not naturally interpreted that way. Typically, a normalized words-documents table is used as an estimator of a words-documents joint probability distribution, where each entry estimates the probability of finding a given word in a given document, whereas the words are assumed to be independent of each other [1, 2]. By contrast, values in a financial data matrix are a general function of stocks and days and in particular might include negative numerical values. Here, the data cannot be regarded as a sample from a joint probability distribution of stocks and days, even if a normalization is applied. Though it can be argued that each entry of the matrix is a sample from a joint probability distribution of three variables: stock, day, and price change - the degenerate nature of this sample must be taken into account: the rate change of a given stock on a given day occurs only once and no statistics exist. Therefore the joint probability distribution of the three variables cannot be estimated

by direct sampling. A similar argument applies to survey data like movie ratings. In this case the sample might be even more degenerate, with many "missing values", since not all the viewers rate all the movies. Although gene expression data can be considered as a repeatable experiment, very often different experimental conditions correspond to only one single column in the matrix. Thus once again a single data point represents the joint statistics of three variables: gene, condition, and expression level.

Nevertheless, in most such cases there is a statistical relationship within rows and/or within columns of a matrix. For instance, people with similar interests typically give similar ratings to movies and movies with similar characteristics give rise to similar rating profiles. Such relationships can be exploited by clustering algorithms that group together similar rows and/or columns, and furthermore make it possible to complete the missing entries in the matrix [3].

The existing clustering techniques can be classified into three major categories. (i) Similarity, or distance-based methods, require a pre-defined similarity measure that can be applied to all data points and possibly to new points as well. The nature of the distance measure is crucial to these techniques and inherently requires expert knowledge of the application domain, which is often unavailable. (ii) Generative modeling techniques in which a specific class of statistical models is chosen to describe the data. As before, an *a priori* choice of an appropriate model is far from obvious for most real world applications. (iii) An alternative line of study, relevant to our work, is the Information Bottleneck (IB) approach [4] and its extensions. Instead of defining the clustering objective through a distortion measure or data generation process, the approach suggests using relevant variables. A tradeoff between compression of the irrelevant and prediction of the relevant variables is then optimized using information theoretic principles. Importantly, the definition of the relevant variable is often natural and obvious for the task at hand, and in turn the method yields the optimal *relevant distortion* for the problem.

Since the original work in [4], multiple studies have highlighted the theoretical and practical importance of the IB method, in particular in the context of cluster analysis [2, 5, 6, 7, 8]. However, the original formulation is based on the availability of co-occurrence data. In practice a given co-occurrence table is treated as a finite sample out of a joint distribution of the variables; namely row and column indices. Unfortunately, as mentioned above, this assumption does do not fit many realistic datasets, thus preventing a direct application of the IB approach in various domains.

To address this issue, in [9] a random walk over the data points is defined, serving to transform non co-occurrence data into a transition probability matrix that can be further analyzed via an IB algorithm. In a more recent work [10] the suggestion is made to use the mutual information between different data points as part of a general information-theoretic treatment to the clustering problem. The resulting algorithm, termed the *Iclust* algorithm, was demonstrated to be superior or comparable to 18 other commonly used clustering techniques over a wide range of applications where the input data cannot be interpreted as co-occurrences [10]. However, both of the approaches have limitations – *Iclust* requires a sufficient amount of columns in the matrix for reliable estimation of the information relations between rows, and the Markovian Relaxation algorithm involves various non-trivial steps in the data pre-process [9].

Here, we suggest an alternative approach, inspired by the *multivariate* IB framework [8]. The multivariate IB principle expands the original IB work to handle situations where multiple systems of clusters are constructed simultaneously with respect to different variables and the input data may correspond to more than two variables. While the multivariate IB was originally proposed for co-occurrence data, we argue here that this framework is rich enough to be rigorously applicable in the new situation. The idea is simple and intuitive: we look for a compact grouping of rows and/or columns such that the product space defined by the resulting clusters is maximally informative about the matrix content, i.e. the matrix entries. We show that this problem can be posed and solved within the original multivariate IB framework. The new choice of relevance variable eliminates the need to know the joint distribution of all the input variables (which is inaccessible in all the applications presented here). Moreover, when missing values are present, the analysis suggests an information theoretic technique for their completion.

We explore the application of this approach to various domains. For gene expression data and financial data we obtain clusters of comparable quality (measured as coherence with manual labeling) to

those obtained by state-of-the-art methods [10]. For movie rating matrix completion, performance is superior to the best known alternatives in the collaborative filtering literature [11].

## 2 Theory

### 2.1 Problem Setting

We henceforth denote the rows of a matrix by $X$, the columns by $Y$, and the matrix entries by $Z$ (and small $x, y$ and $z$ for specific instances). The number of rows is denoted by $n$, and the number of columns by $m$. We regard $X$ and $Y$ as discrete coordinate space and $Z(X, Y)$ as a function. Generalization to higher dimensions and continuous coordinates is readily possible, but not discussed here. For a given matrix, a row $x$, and a column $y$, the value of $z(x, y)$ is assumed to be deterministic. This can be relaxed as well.

The objective is to find "good" partitions of the $X$-$Y$ space that will be informative with respect to the function values $Z(X, Y)$. The partitions are defined by grouping of rows into clusters of rows $C$ and grouping of columns into clusters of columns $D$. The complexity of such partitions is measured by the sum of the weighted mutual information values $nI(X; C) + mI(Y; D)$. For the hard partitions considered in the paper this sum is the number of bits required to describe the partition (see [12]). The informativeness of the partition is measured by another mutual information, $I(C, D; Z)$. In these terms, the goal is to find minimally complex partitions that preserve a given information level about the matrix values $Z$. This can be expressed via the minimization of the following functional:

$$\min_{q(c|x), q(d|y)} nI(X; C) + mI(Y; D) - \beta I(C, D; Z), \tag{1}$$

where $q(c|x)$ is the mapping of rows $x$ to row clusters $c$, $q(d|y)$ is the mapping of columns $y$ to column clusters $d$, and $\beta$ is a Lagrange multiplier controlling the tradeoff between compression and accuracy.

We first derive the relations between the quantities in the above optimization problem and then describe a sequential algorithm for its minimization. We will stick to the following notation conventions: $p$ is used for distributions that involve only input parameters and hence do not change during the analysis, $\hat{p}$ is used for empirical distributions, $q$ for the sought mapping distributions and $\hat{q}$ for empirical distributions dependent on the sought mappings. By the definition of the mutual information [12]:

$$I(X; C) = \sum_{x,c} p(x)q(c|x) \log \frac{q(c|x)}{q(c)} \approx \sum_{x,c} \hat{p}(x)q(c|x) \log \frac{q(c|x)}{\hat{q}(c)}.$$

We define the indicator function:

$$1_{x,y} = \begin{cases} 1, & if\ the\ entry\ (x, y)\ is\ present\ in\ the\ matrix \\ 0, & if\ the\ entry\ (x, y)\ is\ absent\ in\ the\ matrix \end{cases}$$

and denote the total number of populated entries (which is our sample size) by: $N = \sum_{x,y} 1_{x,y}$. Then:

$$\hat{p}(x) = \frac{\sum_y 1_{x,y}}{N} = \frac{Number\ of\ populated\ entries\ in\ row\ x}{Total\ number\ of\ populated\ entries},$$

$$\hat{q}(c) = \sum_x \hat{p}(x)q(c|x),$$

$I(Y; D)$, $\hat{p}(y)$, and $\hat{q}(d)$ are defined similarly.

$$I(C, D; Z) = \sum_{c,d,z} q(c, d)q(z|c, d) \log \frac{q(z|c, d)}{p(z)} \approx \sum_{c,d,z} \hat{q}(c, d)\hat{q}(z|c, d) \log \frac{\hat{q}(z|c, d)}{\hat{p}(z)}.$$

We assume $Z$ is a categorical variable, thus:

$$\hat{p}(z) = \frac{\sum_{x,y:z(x,y)=z} 1}{N} = \frac{Number\ of\ entries\ equal\ to\ z}{Total\ number\ of\ populated\ entries},$$

$$\hat{q}(c,d) = \frac{\sum_{x,y} q(c|x)q(d|y)1_{x,y}}{N} = \frac{Number\ of\ populated\ entries\ in\ section\ c,d}{Total\ number\ of\ populated\ entries},$$

$$\hat{q}(z|c,d) = \frac{\sum_{x,y:z(x,y)=z} q(c|x)q(d|y)}{\sum_{x,y} q(c|x)q(d|y)1_{x,y}} = \frac{Number\ of\ entries\ equal\ to\ z\ in\ section\ c,d}{Number\ of\ populated\ entries\ in\ section\ c,d}.$$

In the special case of complete data matrices $1_{x,y}$ is identically 1 and $\hat{q}(c,d)$ may be decomposed as: $\hat{q}(c,d) = \hat{q}(c)\hat{q}(d)$. In addition $\hat{p}(x)$ and $\hat{p}(y)$ accept the form $\hat{p}(x) = \frac{1}{n}$ and $\hat{p}(y) = \frac{1}{m}$. But in the general case considered in this paper $X$ and $Y$ (and thus $C$ and $D$) are not independent.

## 2.2 Sequential Optimization

Given $q(c|x)$ and $q(d|y)$, one can calculate all the quantities defined above, and in particular the minimization functional $\mathcal{L}_{min} = nI(X;C) + mI(Y;D) - \beta I(C,D;Z)$ defined in equation (1). To minimize $\mathcal{L}_{min}$ (using hard partitions) we can use the sequential (greedy) optimization algorithm suggested in [13]. This algorithm is quite simple:

1. Start with a random (hard) partition $q(c|x)$, $q(d|y)$.
2. Iteratively until convergence (no changes at step (b) are done) traverse all rows $x$ and columns $y$ of a matrix in a random order. For each row/column:
   (a) Draw $x$ (or $y$) from its cluster.
   (b) Reassign it to a new cluster $c^*$ (or $d^*$), so that $\mathcal{L}_{min}$ is minimized. The new cluster may appear to be the old cluster, and then no change is counted.

Due to monotonic decrease in $\mathcal{L}_{min}$, which is lower bounded by $-\beta H(Z)$ the algorithm is guaranteed to converge to some local minima of (1). Multiple random initializations may be used to improve the result. This simple algorithm is by far not the only way to optimize (1), but in practice it was shown to achieve very good results on similar optimization problems [2].

The complexity of the algorithm is analyzed in the complementary material, where it is shown to be $O(M(n+m)|C||D|)$, when $M$ is the number of iterations required for convergence (usually 10-40) and $|C|$, $|D|$ are cardinalities of the corresponding variables.

## 2.3 Minimal Description Length (MDL) Formulation

The minimization functional $\mathcal{L}_{min}$ has three free parameters that have to be externally determined: the tradeoff (or resolution) parameter $\beta$, and the cardinalities $|C|$, and $|D|$. Whereas in some applications they may be given (e.g. the desired number of clusters), there are cases when they also require optimization (as in the example of matrix completion in the next section). To perform such optimization, the Minimum Description Length (MDL) principle [14] is used. The idea behind MDL is that models achieving better compression of the training data - when the compression includes a model description - also achieve better generalization on the test data.

The following compression scheme is defined: $|C|$ row and $|D|$ column clusters define $|C||D|$ sections each getting roughly $\frac{N}{|C||D|}$ samples. The corresponding distributions $\hat{q}(z|c,d)$ over categorical variable $Z$ may be described by $\frac{|Z||C||D|}{2} \log \frac{N}{|C||D|}$ bits (see [14]). As already mentioned, the matrix partition itself may be described by $nI(X;C) + mI(Y;D)$ bits. And given the partition and the distributions $\hat{q}(z|c,d)$ the number of bits required to code the matrix entries is $NH(Z|C,D)$ [12]. Thus the total description length is $nI(X;C) + mI(Y;D) + NH(Z|C,D) + \frac{|Z||C||D|}{2} \log \frac{N}{|C||D|}$. Since $H(Z|C,D) = H(Z) - I(C,D;Z)$ and $H(Z)$ is constant the latter can be omitted from optimization, which results in total minimization functional

$$\mathcal{F}_{mdl} = nI(X;C) + mI(Y;D) - NI(C,D;Z) + \frac{|Z||C||D|}{2} \log \frac{N}{|C||D|}. \tag{2}$$

Observe that constrained on $|C|$, and $|D|$, $\mathcal{L}_{min}$ corresponding to $\mathcal{F}_{mdl}$ accepts the form of

$$\mathcal{L}_{min} = nI(X;C) + mI(Y;D) - NI(C,D;Z), \tag{3}$$

i.e. the optimal tradeoff $\beta = N$ is uniquely determined. Since in practice $\mathcal{F}_{mdl}$ is roughly convex in both $C$ and $D$, the optimal values for these two parameters may be easily determined by scanning.

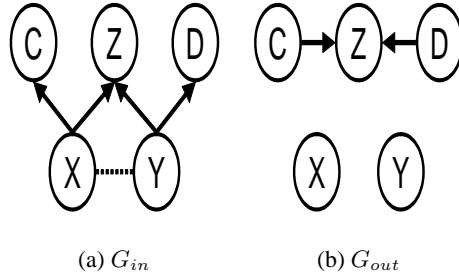

(a) $G_{in}$        (b) $G_{out}$

Figure 1: $G_{in}$ and $G_{out}$ in Multivariate IB formulation.

## 2.4 Relation with the Multivariate Information Bottleneck

Multivariate Information Bottleneck (IB) [8] is an unsupervised approach for structured data exploration. Its core lies in combining the Bayesian networks formalism [15] with the Information Bottleneck method [4]. Multivariate IB searches for a meaningful structured partition of the data, defined by compression variables (in our case these are $C$ and $D$). Two graphs, $G_{in}$ and $G_{out}$, are defined. The former specifies the relations between the input (data) variables – in our case, $X$, $Y$, and $Z$ – and the compression variables. The latter specifies the information terms that are to be preserved by the partition. A tradeoff between the multi-information preserved in the input structure $I^{G_{in}}$ (which we want to minimize) and the multi-information expressed by the target structure $I^{G_{out}}$ (which we want to maximize) is then optimized. For a set of variables $\mathcal{V} = V_1, .., V_n$, a directed acyclic graph (Bayesian network) $G$, and a joint probability distribution over $\mathcal{V}$, $p(\mathcal{V})$, the multi-information $I^G(\mathcal{V})$ is defined as:

$$I^G[p(\mathcal{V})] = \sum_{i=1}^{n} I(V_i; Pa^G(V_i)),$$

where $Pa^G(V_i)$ are the parents of node $V_i$ in $G$.

The graphs $G_{in}$ and $G_{out}$ corresponding to our case are given in Figure 1. (The dashed link between $X$ and $Y$ in $G_{in}$ appears when missing values are present in the matrix and may be chosen in any direction.) The corresponding optimization functional is:

$$\min_{q(c|x),q(d|y)} I^{G_{in}}(X;Y;Z;C;D) - \beta I^{G_{out}}(X;Y;Z;C;D)$$
$$= \min_{q(c|x),q(d|y)} I(X;Y) + I(X,Y;Z) + I(X;C) + I(Y;D) - \beta I(C,D;Z).$$

By observing that $I(X,Y;Z)$ and $I(X;Y)$ are independent of $q(c|x)$ and $q(d|y)$ we can eliminate them from the above optimization and obtain exactly the optimization functional defined in (1), only with equal weighting of $I(X;C)$ and $I(Y;D)$.

Thus the approach may be seen as a special case of the multivariate IB for the graphs $G_{in}$ and $G_{out}$ defined in Figure 1. An important distinction should be made though: unlike the multivariate IB we do not require the existence of the joint probability distribution $p(x, y, z)$. This is achieved by excluding the term $I(X, Y; Z)$ from the optimization functional.

## 2.5 Relation with Information Based Clustering

A recent information-theoretic approach for cluster analysis is given in [10], and is known as information based clustering, abbreviated as *Iclust*. In contrast to the original IB method, *Iclust* is equally applicable to co-occurrence as well as non co-occurrence data. In the following we highlight the relation of our work and this earlier contribution.

By changing the notations used in [10] to those used here we can write the similarity measure $s(x_1, x_2)$ used in [10] as:

$$s(x_1, x_2) = \sum_{z_1, z_2} \sum_{y} p(y)p(z_1, z_2|x_1, x_2, y) \log \frac{p(y)p(z_1, z_2|x_1, x_2, y)}{\sum_{y_1} p(y_1)p(z_1|x_1, y_1) \sum_{y_2} p(y_2)p(z_2|x_2, y_2)}$$

Table 1: **Clusters coherence for the ESR and S&P stock datasets.** The table provides the coherence of the achieved solutions for $N_c = 5, 10, 15$ and 20 row clusters. The results achieved by the Iclust algorithm at the same settings are shown in brackets alongside the results of our algorithm. For the ESR data an average coherence according to the three GOs is shown. Separate results for each GO are provided in the supplementary material[1].

| Dataset | $N_c = 5$ | $N_c = 10$ | $N_c = 15$ | $N_c = 20$ |
|---------|-----------|------------|------------|------------|
| ESR | 69 (79) | 53 (49) | 50 (52) | 42 (42) |
| S&P | 94 (88) | 83 (91) | 92 (93) | 86 (86) |

$$= I(Z_1; Z_2 | x_1, x_2).$$

Substituting this in the optimization functional of [10], changing maximization to minimization by flipping the sign, and substituting $T = \frac{1}{\beta}$ we obtain:

$$\min_{q(c|x)} I(X; C) - \beta \sum_c q(c) \sum_{x_1, x_2} q(x_1|c) q(x_2|c) s(x_1, x_2)$$

$$= \min_{q(c|x)} I(X; C) - \beta \sum_{c, x_1, x_2} q(c, x_1, x_2) I(Z_1; Z_2 | x_1, x_2),$$

which is reminiscent of equation (1) if no column ($Y$) grouping is done and cluster variance is measured through pairwise distances and not based on a centroid model.

Importantly, in order to be able to evaluate $I(Z_1; Z_2 | x_1, x_2)$, I-Clust requires a sufficient amount of columns to be available, whereas our approach can operate with any amount of columns given. Alternately, even when the data contain many columns, but are relatively sparse, *i.e.*, have many missing values, evaluating $I(Z_1; Z_2 | x_1, x_2)$ might be prohibitive as it requires a large enough intersection of non-missing observations for $z_1$ and $z_2$. In our approach it is not a limitation. On the contrary, the approach is designed to cope with this kind of data and resolves the problem by simultaneous grouping of rows and columns of the matrix to amplify statistics.

## 3   Applications

We first compare our algorithm to I-Clust, as it was shown to be superior/comparable to 18 other commonly used clustering techniques over a wide range of application domains [10]. We then describe an experiment on matrix completion. Another application to a small dataset is provided in the supplementary material[1]. In the last two cases Iclust is not directly applicable. The multivariate IB is not directly applicable to all the provided examples.

### 3.1   One Dimensional Clustering - Comparison to I-Clust

We focus on two applications reported in [10]. For purposes of comparison we restrict our algorithm to cluster only the rows dimension of the matrix by setting the number of column clusters, $|D|$, equal to the number of columns, $m$. This simplifies the objective functional defined in equation (1) to $\mathcal{L}_{min} = I(X; C) - \beta I(C, Y; Z)$. (To have a similar form to [10] we incorporate factor $n$ multiplying $I(X; C)$ in $\beta$.) For both applications we use exactly the same setting as [10], including row-wise quantization of the input data into five equally populated bins and choosing the same values for the $\beta$ parameter.

The first dataset consists of gene expression levels of yeast genes in 173 various forms of environmental stress [16]. Previous analysis identified a group of $\approx 300$ stress-induced and $\approx 600$ stress-repressed genes with "nearly identical but opposite patterns of expression in response to the environmental shifts" [17]. These 900 genes were termed the yeast environmental stress response (ESR) module. Following [10] we cluster the genes into $|C| = 5, 10, 15$, and 20 clusters. To assess

the biological significance of the results we consider the *coherence* [18] of the obtained clusters with respect to three Gene Ontologies (GOs) [19]. Specifically, the coherence of a cluster is defined as the percentage of elements within this cluster that are given an annotation that was found to be significantly enriched in the cluster [18]. The results achieved by our algorithm on this dataset are comparable to the results achieved by I-Clust in all the verified settings - see Table 1.

The second dataset is the day-to-day fractional changes in the price of the stocks in the Standard & Poor (S&P) 500 list[2], during 273 trading days of 2003. As with the gene expression data we take exactly the same setting used by [10] and cluster the stocks into $|C| = 5, 10, 15$ and $20$ clusters. To evaluate the coherence of the ensuing clusters we use the Global Industry Classification Standard[3], which classifies companies into four different levels, organized in a hierarchical tree: sector, industry group, industry, and subindustry. As with the ESR dataset our results are comparable with the results of I-Clust for all the configurations - see Table 1.

## 3.2 Matrix Completion and Collaborative Filtering

Here, we explore the full power of our algorithm in simultaneous grouping of rows and columns of a matrix. A highly relevant application is matrix completion - given a matrix with missing values we would like to be able to complete it by utilizing similarities between rows and columns. This problem is at the core of collaborative filtering applications, but may also appear in other fields. We test our algorithm on the publicly available MovieLens 100K dataset[4]. The dataset consists of 100,000 ratings on a five-star scale for 1,682 movies by 943 users. We take the five non-overlapping splits of the dataset into 80% train on 20% test size provided at the MovieLens web site. We stress that with this division the training data are extremely sparse - only 5% of the training matrix entries are populated, whereas 95% of the values are missing.

To find a "good" bi-clustering of the ratings matrix, minimization of $\mathcal{F}_{mdl}$ defined in (2) is done by scanning cluster cardinalities $|C|$ and $|D|$ and optimizing $\mathcal{L}_{min}$ as defined in (3) for each fixed pair of $|C|, |D|$. The minimum of $\mathcal{F}_{mdl}$ is obtained at $|C| \approx 13$ and $|D| \approx 6$ with beyond 1% sensitivity to small changes in $|C|$ and in $|D|$ both in $\mathcal{F}_{mdl}$ values and in prediction accuracy. See supplementary material[1] for visualization of the solution at $|C| = 4$ and $|D| = 3$.

To measure the accuracy of our algorithm we use mean absolute error (MAE) metrics, which is commonly used for evaluation on this dataset [11]. The mean absolute error is defined as: $MAE = \frac{1}{N} \sum_{i=1}^{N} |z_i - r_i|$, where $z_i$-s are the predicted and $r_i$-s are the actual ratings. To convert the distributions $\hat{q}(z|c,d)$ we obtained in our clustering procedure to concrete predictions we take the median of $z$ values within each section $c, d$.

Note that our algorithm is general and does not directly optimize the MAE error functional. Nevertheless we obtain 0.72 MAE (with a deviation of less than 0.01 over multiple experiments). This confidently beats the "magic barrier" of 0.73 reported in the collaborative filtering literature [11].

The root mean squared error (RMSE) measured for the same clustering with a mean of $z$ values within each section $c, d$ taken for prediction yields 0.96 (with a deviation below 0.01). This is much better than 1.165 RMSE reported for a dataset 20 times larger [20] and quite close to 0.9525 RMSE reported by Netflix for a dataset 1000 times larger of a similar nature[5].

## 4 Discussion

A new model independent approach to the analysis of data given in the form of samples of a function $Z(X, Y)$ rather than samples of co-occurrence statistics of $X$ and $Y$ is introduced. From a theoretical viewpoint the approach is a much required extension of the Information Bottleneck method that allows for its application to entirely new domains. The approach also provides a natural way for bi-clustering and matrix completion. From a practical viewpoint the major contribution of the paper is the achievement of the best known results for a wide range of applications with a single algorithm. As well, we improve on the results of prediction of missing values (collaborative filtering).

Possible directions for further research include generalization to continuous data values, such as those obtained in gene expression and stock price data, and relaxation of the algorithm to "soft" clustering solutions. Another interesting extension would be to dimensionality reduction, rather than clustering, as occurs in IB when applied to continuous variables [21].

The proposed framework also provides a natural platform for derivation of generalization bounds for missing values prediction that will be discussed elsewhere.

## Footnotes

[1]Supplementary material is available at http://www.cs.huji.ac.il/~seldin

[2]Available at http://www.standardpoors.com

[3]Available at http://wrds.wharton.upenn.edu

[4]Available at http://www.grouplens.org

[5]See http://www.netflixprize.com/rules

## References

[1] Noan Slonim and Naftali Tishby. Document clustering using word clusters via the information bottleneck method. In *Proceedings of 23rd Annual International ACM SIGIR Conference on Research and Development in Information Retrieval*, 2000.

[2] Noam Slonim. *The Information Bottleneck: Theory and Applications*. PhD thesis, The Hebrew University of Jerusalem, 2002.

[3] Sara C. Madeira and Arlindo L. Oliveira. Biclustering algorithms for biological data analysis: A survey. *IEEE/ACM Transactions on Computational Biology and Bioinformatics*, 1(1):24–45, January 2004.

[4] Naftali Tishby, Fernando Pereira, and William Bialek. The information bottleneck method. In *Allerton Conference on Communication, Control and Computation*, volume 37, pages 368–379. 1999.

[5] Janne Sinkkonen and Samuel Kaski. Clustering based on conditional distributions in an auxiliary space. *Neural Computation*, 14(1):217–239, 2002.

[6] David Gondek and Thomas Hofmann. Non-redundant data clustering. In *4th IEEE International Conference on Data Mining*, 2004.

[7] Susanne Still, William Bialek, and Léon Bottou. Geometric clustering using the information bottleneck method. In *Advances in Neural Information Processing Systems 16*.

[8] Noam Slonim, Nir Friedman, and Naftali Tishby. Multivariate information bottleneck. *Neural Computation*, 18, 2006.

[9] Naftali Tishby and Noam Slonim. Data clustering by markovian relaxation and the information bottleneck method. In *NIPS*, 2000.

[10] Noam Slonim, Gurinder Singh Atwal, Gasper Tracik, and William Bialek. Information-based clustering. In *Proceedings of the National Academy of Science (PNAS)*, volume 102, pages 18297–1830, Dec. 2005.

[11] J. Herlocker, J. Konstan, L. Terveen, and J. Riedl. Evaluating collaborative filtering recommender systems. In *ACM Transactions on Information Systems*, volume 22(1), pages 5–53, January 2004.

[12] Thomas M. Cover and Joy A. Thomas. *Elements of Information Theory*. Wiley Series in Telecommunications. John Wiley & Sons, New York, NY, 1991.

[13] Noam Slonim, Nir Friedman, and Naftali Tishby. Unsupervised document classification using sequential information maximization. In *Proceedings of the $25^{th}$ Annual International ACM SIGIR Conference on Research and Development in Information Retrieval (SIGIR)*, 2002.

[14] A. Barron, J. Rissanen, and B. Yu. The minimum description length principle in coding and modeling. *IEEE Trans. Info. Theory*, 44:2743–2760, 1998.

[15] J. Pearl. *Probabilistic Reasoning in Intelligent Systems: Networks of Plausible Inference*. San Mateo, CA: Morgan Kaufman Publishers, 1988.

[16] Gasch A.P., Spellman P.T., Kao C.M., Carmel-Harel O., Eisen M.B., Storz G., Botstein D., and Brown P.O. Genomic expression programs in the response of yeast cells to environmental changes. *Molecular Biology. Cell*, 11(12):4241–57, December 2000.

[17] A.P. Gasch. The environmental stress response: a common yeast response to environmental stresses. *Topics in Current Genetics (series editor S. Hohmann)*, 1:11–70, 2002.

[18] Segal E., Shapira M., Regev A., Pe'er D., Botstein D., Koller D., and Friedman N. Module networks: identifying regulatory modules and their condition-specific regulators from gene expression data. *Natural Genetics*, 34(2):166–76, 2003.

[19] M. Ashburner, C. A. Ball, J. A. Blake, D. Botstein, H. Butler, J. M. Cherry, A. P. Davis, K. Dolinski, S. S. Dwight, J. T. Eppig, M. A. Harris, D. P. Hill, L. Issel-Tarver, A. Kasarskis, S. Lewis, J. C. Matese, J. E. Richardson, M. Ringwald, G. M. Rubin, and G. Sherlock. Gene ontology: tool for the unification of biology. *Nature Genetics*, 25:25–29, May 2000.

[20] Thomas Hoffman. Latent semantic models for collaborative filtering. In *ACM Transactions on Information Systems*, volume 22, January 2004.

[21] Gal Chechik, Amir Globerson, Naftali Tishby, and Yair Weiss. Gaussian information bottleneck. *Journal of Machine Learning Research*, 6:165–188, January 2005.
